# The Unscented Particle Filter

**Rudolph van der Merwe**
Oregon Graduate Institute
Electrical and Computer Engineering
P.O. Box 91000,Portland,OR 97006, USA
rvdmerwe@ece.ogi.edu

**Arnaud Doucet**
Cambridge University
Engineering Department
Cambridge CB2 1PZ, England
ad2@eng.cam.ac.uk

**Nando de Freitas**
UC Berkeley, Computer Science
387 Soda Hall, Berkeley
CA 94720-1776 USA
jfgf@cs.berkeley.edu

**Eric Wan**
Oregon Graduate Institute
Electrical and Computer Engineering
P.O. Box 91000,Portland,OR 97006, USA
ericwan@ece.ogi.edu

## Abstract

In this paper, we propose a new particle filter based on sequential importance sampling. The algorithm uses a bank of unscented filters to obtain the importance proposal distribution. This proposal has two very "nice" properties. Firstly, it makes efficient use of the latest available information and, secondly, it can have heavy tails. As a result, we find that the algorithm outperforms standard particle filtering and other nonlinear filtering methods very substantially. This experimental finding is in agreement with the theoretical convergence proof for the algorithm. The algorithm also includes resampling and (possibly) Markov chain Monte Carlo (MCMC) steps.

## 1 Introduction

Filtering is the problem of estimating the states (parameters or hidden variables) of a system as a set of observations becomes available on-line. This problem is of paramount importance in many fields of science, engineering and finance. To solve it, one begins by modelling the evolution of the system and the noise in the measurements. The resulting models typically exhibit complex nonlinearities and non-Gaussian distributions, thus precluding analytical solution.

The best known algorithm to solve the problem of non-Gaussian, nonlinear filtering (filtering for short) is the extended Kalman filter (Anderson and Moore 1979). This filter is based upon the principle of linearising the measurements and evolution models using Taylor series expansions. The series approximations in the EKF algorithm can, however, lead to poor representations of the nonlinear functions and probability distributions of interest. As as result, this filter can diverge.

Recently, Julier and Uhlmann (Julier and Uhlmann 1997) have introduced a filter founded on the intuition that it is easier to approximate a Gaussian distribution

than it is to approximate arbitrary nonlinear functions. They named this filter the unscented Kalman filter (UKF). They have shown that the UKF leads to more accurate results than the EKF and that in particular it generates much better estimates of the covariance of the states (the EKF seems to underestimate this quantity). The UKF has, however, the limitation that it does not apply to general non-Gaussian distributions.

Another popular solution strategy for the general filtering problem is to use sequential Monte Carlo methods, also known as particle filters (PFs): see for example (Doucet, Godsill and Andrieu 2000, Doucet, de Freitas and Gordon 2001, Gordon, Salmond and Smith 1993). These methods allow for a complete representation of the posterior distribution of the states, so that any statistical estimates, such as the mean, modes, kurtosis and variance, can be easily computed. They can therefore, deal with any nonlinearities or distributions.

PFs rely on importance sampling and, as a result, require the design of proposal distributions that can approximate the posterior distribution reasonably well. In general, it is hard to design such proposals. The most common strategy is to sample from the probabilistic model of the states evolution (transition prior). This strategy can, however, fail if the new measurements appear in the tail of the prior or if the likelihood is too peaked in comparison to the prior. This situation does indeed arise in several areas of engineering and finance, where one can encounter sensors that are very accurate (peaked likelihoods) or data that undergoes sudden changes (non-stationarities): see for example (Pitt and Shephard 1999, Thrun 2000). To overcome this problem, several techniques based on linearisation have been proposed in the literature (de Freitas 1999, de Freitas, Niranjan, Gee and Doucet 2000, Doucet et al. 2000, Pitt and Shephard 1999). For example, in (de Freitas et al. 2000), the EKF Gaussian approximation is used as the proposal distribution for a PF. In this paper, we follow the same approach, but replace the EKF proposal by a UKF proposal. The resulting filter should perform better not only because the UKF is more accurate, but because it also allows one to control the rate at which the tails of the proposal distribution go to zero. It becomes thus possible to adopt heavier tailed distributions as proposals and, consequently, obtain better importance samplers (Gelman, Carlin, Stern and Rubin 1995). Readers are encouraged to consult our technical report for further results and implementation details (van der Merwe, Doucet, de Freitas and Wan 2000)[1].

## 2  Dynamic State Space Model

We apply our algorithm to general state space models consisting of a transition equation $p(\mathbf{x}_t|\mathbf{x}_{t-1})$ and a measurement equation $p(\mathbf{y}_t|\mathbf{x}_t)$. That is, the states follow a Markov process and the observations are assumed to be independent given the states. For example, if we are interested in nonlinear, non-Gaussian regression, the model can be expressed as follows

$$\begin{aligned}
\mathbf{x}_t &= \mathbf{f}(\mathbf{x}_{t-1}, \mathbf{v}_{t-1}) \\
\mathbf{y}_t &= \mathbf{h}(\mathbf{u}_t, \mathbf{x}_t, \mathbf{n}_t)
\end{aligned}$$

where $\mathbf{u}_t \in \mathbb{R}^{n_u}$ denotes the input data at time $t$, $\mathbf{x}_t \in \mathbb{R}^{n_x}$ denotes the states (or parameters) of the model, $\mathbf{y}_t \in \mathbb{R}^{n_y}$ the observations, $\mathbf{v}_t \in \mathbb{R}^{n_v}$ the process noise and $\mathbf{n}_t \in \mathbb{R}^{n_n}$ the measurement noise. The mappings $\mathbf{f} : \mathbb{R}^{n_x} \times \mathbb{R}^{n_v} \mapsto \mathbb{R}^{n_x}$ and $\mathbf{h} : (\mathbb{R}^{n_x} \times \mathbb{R}^{n_u}) \times \mathbb{R}^{n_n} \mapsto \mathbb{R}^{n_y}$ represent the deterministic process and measurement models. To complete the specification of the model, the prior distribution (at $t = 0$)

is denoted by $p(\mathbf{x}_0)$. Our goal will be to approximate the posterior distribution $p(\mathbf{x}_{0:t}|\mathbf{y}_{1:t})$ and one of its marginals, the filtering density $p(\mathbf{x}_t|\mathbf{y}_{1:t})$, where $\mathbf{y}_{1:t} = \{\mathbf{y}_1, \mathbf{y}_2, \ldots, \mathbf{y}_t\}$. By computing the filtering density recursively, we do not need to keep track of the complete history of the states.

## 3  Particle Filtering

Particle filters allow us to approximate the posterior distribution $p(\mathbf{x}_{0:t}|\mathbf{y}_{1:t})$ using a set of $N$ weighted samples (particles) $\left\{\mathbf{x}_{0:t}^{(i)}; i = 1, ..., N\right\}$, which are drawn from an importance proposal distribution $q(\mathbf{x}_{0:t}|\mathbf{y}_{1:t})$. These samples are propagated in time as shown in Figure 1. In doing so, it becomes possible to map intractable integration problems (such as computing expectations and marginal distributions) to easy summations. This is done in a rigorous setting that ensures convergence according to the strong law of large numbers

$$\frac{1}{N} \sum_{i=1}^{N} f_t\left(\mathbf{x}_{0:t}^{(i)}\right) \xrightarrow[N \to +\infty]{a.s.} \int f_t\left(\mathbf{x}_{0:t}\right) P\left(d\mathbf{x}_{0:t}|\mathbf{y}_{1:t}\right)$$

where $\xrightarrow{a.s.}$ denotes almost sure convergence and $f_t : \mathbb{R}^{n_x} \to \mathbb{R}^{n_{f_t}}$ is some function of interest. For example, it could be the conditional mean, in which case $f_t(\mathbf{x}_{0:t}) = \mathbf{x}_{0:t}$, or the conditional covariance of $\mathbf{x}_t$ with $f_t(\mathbf{x}_{0:t}) = \mathbf{x}_t \mathbf{x}_t' -$

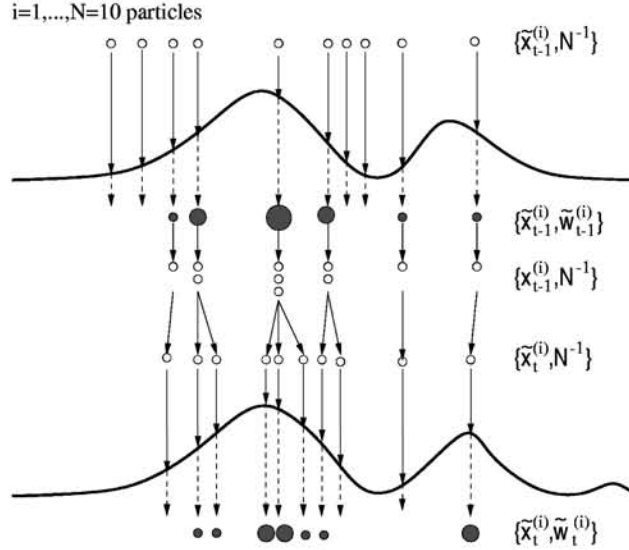

Figure 1: In this example, a particle filter starts at time $t-1$ with an unweighted measure $\{\widetilde{\mathbf{x}}_{t-1}^{(i)}, N^{-1}\}$, which provides an approximation of $p(\mathbf{x}_{t-1}|\mathbf{y}_{1:t-2})$. For each particle we compute the importance weights using the information at time $t-1$. This results in the weighted measure $\{\widetilde{\mathbf{x}}_{t-1}^{(i)}, \widetilde{w}_{t-1}^{(i)}\}$, which yields an approximation $p(\mathbf{x}_{t-1}|\mathbf{y}_{1:t-1})$. Subsequently, a resampling step selects only the "fittest" particles to obtain the unweighted measure $\{\widetilde{\mathbf{x}}_{t-1}^{(i)}, N^{-1}\}$, which is still an approximation of $p(\mathbf{x}_{t-1}|\mathbf{y}_{1:t-1})$. Finally, the sampling (prediction) step introduces variety, resulting in the measure $\{\widetilde{\mathbf{x}}_t^{(i)}, N^{-1}\}$.

$\mathbb{E}_{p(\mathbf{x}_t|\mathbf{y}_{1:t})}[\mathbf{x}_t] \mathbb{E}'_{p(\mathbf{x}_t|\mathbf{y}_{1:t})}[\mathbf{x}_t]$. A Generic PF algorithm involves the following steps.

---

### Generic PF

1. Underline{Sequential importance sampling step}
   - For $i = 1, \ldots, N$, sample $\widetilde{\mathbf{x}}_t^{(i)} \sim q(\mathbf{x}_t|\mathbf{x}_{0:t-1}^{(i)}, \mathbf{y}_{1:t})$ and update the trajectories $\widetilde{\mathbf{x}}_{0:t}^{(i)} \triangleq \left(\widetilde{\mathbf{x}}_t^{(i)}, \mathbf{x}_{0:t-1}^{(i)}\right)$
   - For $i = 1, \ldots, N$, evaluate the importance weights up to a normalizing constant:

   $$w_t^{(i)} = \frac{p(\widetilde{\mathbf{x}}_{0:t}^{(i)}|\mathbf{y}_{1:t})}{q(\widetilde{\mathbf{x}}_t^{(i)}|\mathbf{x}_{0:t-1}^{(i)}, \mathbf{y}_{1:t}) p(\widetilde{\mathbf{x}}_{0:t-1}^{(i)}|\mathbf{y}_{1:t-1})}$$

   - For $i = 1, \ldots, N$, normalize the weights: $\quad \widetilde{w}_t^{(i)} = w_t^{(i)} \left[\sum_{j=1}^{N} w_t^{(j)}\right]^{-1}$.

2. Underline{Selection step}
   - Multiply/ suppress samples $(\widetilde{\mathbf{x}}_{0:t}^{(i)})$ with high/low importance weights $\widetilde{w}_t^{(i)}$, respectively, to obtain $N$ random samples $(\bar{\mathbf{x}}_{0:t}^{(i)})$ approximately distributed according to $p(\bar{\mathbf{x}}_{0:t}^{(i)}|\mathbf{y}_{1:t})$.

3. Underline{MCMC step}
   - Apply a Markov transition kernel with invariant distribution given by $p(\mathbf{x}_{0:t}^{(i)}|\mathbf{y}_{1:t})$ to obtain $(\mathbf{x}_{0:t}^{(i)})$.

---

In the above algorithm, we can restrict ourselves to importance functions of the form $q(\mathbf{x}_{0:t}|\mathbf{y}_{1:t}) = q(\mathbf{x}_0) \prod_{k=1}^{t} q(\mathbf{x}_k|\mathbf{y}_{1:k}, \mathbf{x}_{1:k-1})$ to obtain a recursive formula to evaluate the importance weights

$$w_t \propto \frac{p(\mathbf{y}_t|\mathbf{y}_{1:t-1}, \mathbf{x}_{0:t}) p(\mathbf{x}_t|\mathbf{x}_{t-1})}{q(\mathbf{x}_t|\mathbf{y}_{1:t}, \mathbf{x}_{1:t-1})}$$

There are infinitely many possible choices for $q(\mathbf{x}_{0:t}|\mathbf{y}_{1:t})$, the only condition being that its support must include that of $p(\mathbf{x}_{0:t}|\mathbf{y}_{1:t})$. The simplest choice is to just sample from the prior, $p(\mathbf{x}_t|\mathbf{x}_{t-1})$, in which case the importance weight is equal to the likelihood, $p(\mathbf{y}_t|\mathbf{y}_{1:t-1}, \mathbf{x}_{0:t})$. This is the most widely used distribution, since it is simple to compute, but it can be inefficient, since it ignores the most recent evidence, $\mathbf{y}_t$.

The selection (resampling) step is used to eliminate the particles having low importance weights and to multiply particles having high importance weights (Gordon et al. 1993). This is done by mapping the weighted measure $\{\widetilde{\mathbf{x}}_t^{(i)}, \widetilde{w}_t^{(i)}\}$ to an unweighted measure $\{\widetilde{\mathbf{x}}_t^{(i)}, N^{-1}\}$ that provides an approximation of $p(\mathbf{x}_t|\mathbf{y}_{1:t})$. After the selection scheme at time $t$, we obtain $N$ particles distributed marginally approximately according to $p(\mathbf{x}_{0:t}|\mathbf{y}_{1:t})$. One can, therefore, apply a Markov kernel (for example, a Metropolis or Gibbs kernel) to each particle and the resulting distribution will still be $p(\mathbf{x}_{0:t}|\mathbf{y}_{1:t})$. This step usually allows us to obtain better results and to treat more complex models (de Freitas 1999).

## 4  The Unscented Particle Filter

As mentioned earlier, using the transition prior as proposal distribution can be inefficient. As illustrated in Figure 2, if we fail to use the latest available information to propose new values for the states, only a few particles might survive. *It is therefore of paramount importance to move the particles towards the regions of high likelihood.* To achieve this, we propose to use the unscented filter as proposal distribution. This simply requires that we propagate the sufficient statistics of the UKF for each particle. For exact details, please refer to our technical report (van der Merwe et al. 2000).

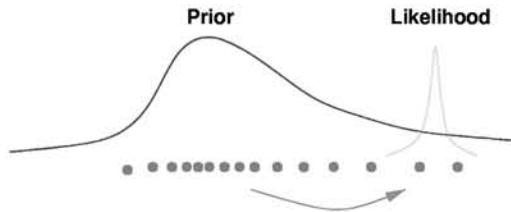

Figure 2: The UKF proposal distribution allows us to move the samples in the prior to regions of high likelihood. This is of paramount importance if the likelihood happens to lie in one of the tails of the prior distribution, or if it is too narrow (low measurement error).

## 5  Theoretical Convergence

Let $B(\mathbb{R}^n)$ be the space of bounded, Borel measurable functions on $\mathbb{R}^n$. We denote $\|f\| \triangleq \sup_{x \in \mathbb{R}^n} |f(x)|$. The following theorem is a straightforward extension of previous results in (Crisan and Doucet 2000).

**Theorem 1** *If the importance weight*

$$w_t \propto \frac{p(\mathbf{y}_t | \mathbf{x}_t) \, p(\mathbf{x}_t | \mathbf{x}_{t-1})}{q(\mathbf{x}_t | \mathbf{x}_{0:t-1}, \mathbf{y}_{1:t})} \tag{1}$$

*is upper bounded for any* $(\mathbf{x}_{t-1}, \mathbf{y}_t)$*, then, for all* $t \geq 0$*, there exists* $c_t$ *independent of* $N$*, such that for any* $f_t \in B\left(\mathbb{R}^{n_x \times (t+1)}\right)$

$$\mathbb{E}\left[\left(\frac{1}{N}\sum_{i=1}^{N} f_t\left(\mathbf{x}_{0:t}^{(i)}\right) - \int f_t(\mathbf{x}_{0:t})\, p(d\mathbf{x}_{0:t} | \mathbf{y}_{1:t})\right)^2\right] \leq c_t \frac{\|f_t\|^2}{N}. \tag{2}$$

The expectation in equation 2 is with respect to the randomness introduced by the particle filtering algorithm. This convergence result shows that, under very lose assumptions, convergence of the (unscented) particle filter is ensured and that the convergence rate of the method is independent of the dimension of the state-space. The only crucial assumption is to ensure that $w_t$ is upper bounded, that is that the proposal distribution $q(\mathbf{x}_t | \mathbf{x}_{0:t-1}, \mathbf{y}_{1:t})$ has heavier tails than $p(\mathbf{y}_t | \mathbf{x}_t)\, p(\mathbf{x}_t | \mathbf{x}_{t-1})$. Considering this theoretical result, it is not surprising that the UKF (which has heavier tails than the EKF) can yield better estimates.

# 6 Demonstration

For this experiment, a time-series is generated by the following process model $x_{t+1} = 1 + sin(\omega \pi t) + \phi x_t + v_t$, where $v_t$ is a Gamma(3,2) random variable modeling the process noise, and $\omega = 4e - 2$ and $\phi = 0.5$ are scalar parameters. A non-stationary observation model,

$$y_t = \begin{cases} \phi x_t^2 + n_t & t \leq 30 \\ \phi x_t - 2 + n_t & t > 30 \end{cases}$$

is used. The observation noise, $n_t$, is drawn from a zero-mean Gaussian distribution. Given only the noisy observations, $y_t$, a few different filters were used to estimate the underlying clean state sequence $x_t$ for $t = 1 \ldots 60$. The experiment was repeated 100 times with random re-initialization for each run. All of the particle filters used 200 particles. Table 1 summarizes the performance of the different filters. The

| Algorithm | MSE | |
|---|---|---|
| | mean | var |
| Extended Kalman Filter (EKF) | 0.374 | 0.015 |
| Unscented Kalman Filter (UKF) | 0.280 | 0.012 |
| Particle Filter : generic | 0.424 | 0.053 |
| Particle Filter : MCMC move step | 0.417 | 0.055 |
| Particle Filter : EKF proposal | 0.310 | 0.016 |
| Particle Filter : EKF proposal and MCMC move step | 0.307 | 0.015 |
| Particle Filter : UKF proposal (*"Unscented Particle Filter"*) | 0.070 | 0.006 |
| Particle Filter : UKF proposal and MCMC move step | 0.074 | 0.008 |

Table 1: Mean and variance of the MSE calculated over 100 independent runs.

table shows the means and variances of the mean-square-error (MSE) of the state estimates. Note that MCMC could improve results in other situations. Figure 3 compares the estimates generated from a single run of the different particle filters. The superior performance of the unscented particle filter is clearly evident. Figure

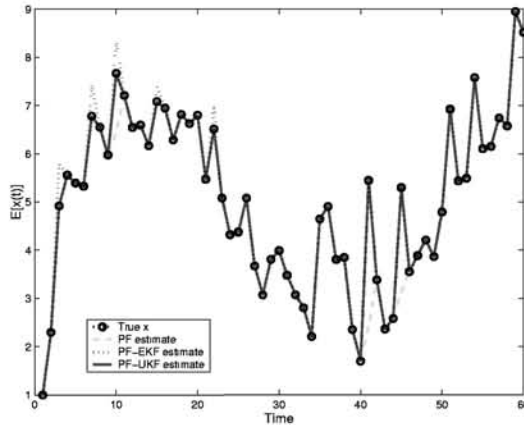

Figure 3: Plot of the state estimates generated by different filters.

4 shows the estimates of the state covariance generated by a stand-alone EKF and UKF for this problem. Notice how the EKF's estimates are consistently smaller than those generated by the UKF. This property makes the UKF better suited than the EKF for proposal distribution generation within the particle filter framework.

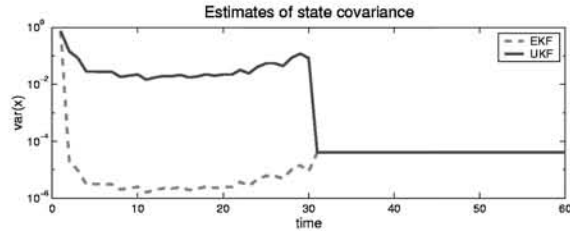

Figure 4: EKF and UKF estimates of state covariance.

# 7    Conclusions

We proposed a new particle filter that uses unscented filters as proposal distributions. The convergence proof and empirical evidence, clearly, demonstrate that this algorithm can lead to substantial improvements over other nonlinear filtering algorithms. The algorithm is well suited for engineering applications, when the sensors are very accurate but nonlinear, and financial time series, where outliers and heavy tailed distributions play a significant role in the analysis of the data. For further details and experiments, please refer to our report (van der Merwe et al. 2000).

## Footnotes

[1]The TR and software are available at `http://www.cs.berkeley.edu/~jfgf`.

# References

Anderson, B. D. and Moore, J. B. (1979). *Optimal Filtering*, Prentice-Hall, New Jersey.

Crisan, D. and Doucet, A. (2000). Convergence of generalized particle filters, *Technical Report CUED/F-INFENG/TR 381*, Cambridge University Engineering Department.

de Freitas, J. F. G. (1999). *Bayesian Methods for Neural Networks*, PhD thesis, Department of Engineering, Cambridge University, Cambridge, UK.

de Freitas, J. F. G., Niranjan, M., Gee, A. H. and Doucet, A. (2000). Sequential Monte Carlo methods to train neural network models, *Neural Computation* **12**(4): 955–993.

Doucet, A., de Freitas, J. F. G. and Gordon, N. J. (eds) (2001). *Sequential Monte Carlo Methods in Practice*, Springer-Verlag.

Doucet, A., Godsill, S. and Andrieu, C. (2000). On sequential Monte Carlo sampling methods for Bayesian filtering, *Statistics and Computing* **10**(3): 197–208.

Gelman, A., Carlin, J. B., Stern, H. S. and Rubin, D. B. (1995). *Bayesian Data Analysis*, Chapman and Hall.

Gordon, N. J., Salmond, D. J. and Smith, A. F. M. (1993). Novel approach to nonlinear/non-Gaussian Bayesian state estimation, *IEE Proceedings-F* **140**(2): 107–113.

Julier, S. J. and Uhlmann, J. K. (1997). A new extension of the Kalman filter to nonlinear systems, *Proc. of AeroSense: The 11th International Symposium on Aerospace/Defence Sensing, Simulation and Controls, Orlando, Florida.*, Vol. Multi Sensor Fusion, Tracking and Resource Management II.

Pitt, M. K. and Shephard, N. (1999). Filtering via simulation: Auxiliary particle filters, *Journal of the American Statistical Association* **94**(446): 590–599.

Thrun, S. (2000). Monte Carlo POMDPs, *in* S. Solla, T. Leen and K.-R. Müller (eds), *Advances in Neural Information Processing Systems 12*, MIT Press, pp. 1064–1070.

van der Merwe, R., Doucet, A., de Freitas, J. F. G. and Wan, E. (2000). The unscented particle filter, *Technical Report CUED/F-INFENG/TR 380*, Cambridge University Engineering Department.
